# Globally Trained Handwritten Word Recognizer using Spatial Representation, Convolutional Neural Networks and Hidden Markov Models

**Yoshua Bengio** *
Dept. Informatique et Recherche Opérationnelle
Université de Montréal
Montreal, Qc H3C-3J7

**Yann Le Cun**
AT&T Bell Labs
Holmdel NJ 07733

**Donnie Henderson**
AT&T Bell Labs
Holmdel NJ 07733

## Abstract

We introduce a new approach for on-line recognition of handwritten words written in unconstrained mixed style. The preprocessor performs a word-level normalization by fitting a model of the word structure using the EM algorithm. Words are then coded into low resolution "annotated images" where each pixel contains information about trajectory direction and curvature. The recognizer is a convolution network which can be spatially replicated. From the network output, a hidden Markov model produces word scores. The entire system is globally trained to minimize word-level errors.

## 1 Introduction

Natural handwriting is often a mixture of different "styles", lower case printed, upper case, and cursive. A reliable recognizer for such handwriting would greatly improve interaction with pen-based devices, but its implementation presents new

technical challenges. Characters taken in isolation can be very ambiguous, but considerable information is available from the context of the whole word. We propose a word recognition system for pen-based devices based on four main modules: a preprocessor that normalizes a word, or word group, by fitting a geometrical model to the word structure using the EM algorithm; a module that produces an "annotated image" from the normalized pen trajectory; a replicated convolutional neural network that spots and recognizes characters; and a Hidden Markov Model (HMM) that interprets the networks output by taking word-level constraints into account. The network and the HMM are *jointly* trained to minimize an error measure defined at the word level.

Many on-line handwriting recognizers exploit the sequential nature of pen trajectories by representing the input in the time domain. While these representations are compact and computationally advantageous, they tend to be sensitive to stroke order, writing speed, and other irrelevant parameters. In addition, global geometric features, such as whether a stroke crosses another stroke drawn at a different time, are not readily available in temporal representations. To avoid this problem we designed a representation, called AMAP, that preserves the pictorial nature of the handwriting.

In addition to recognizing characters, the system must also correctly segment the characters within the words. One approach, that we call INSEG, is to recognize a large number of heuristically segmented candidate characters and combine them optimally with a postprocessor (Burges et al 92, Schenkel et al 93). Another approach, that we call OUTSEG, is to delay all segmentation decisions until after the recognition, as is often done in speech recognition. An OUTSEG recognizer must accept entire words as input and produce a sequence of scores for each character at each location on the input. Since the word normalization cannot be done perfectly, the recognizer must be robust with respect to relatively large distortions, size variations, and translations. An elastic word model –e.g., an HMM– can extract word candidates from the network output. The HMM models the long-range sequential structure while the neural network spots and classifies characters, using local spatial structure.

## 2   Word Normalization

Input normalization reduces intra-character variability, simplifying character recognition. This is particularly important when recognizing entire words. We propose a new word normalization scheme, based on fitting a geometrical model of the word structure. Our model has four "flexible" lines representing respectively the ascenders line, the core line, the base line and the descenders line (see Figure 1). Points on the lines are parameterized as follows:

$$\mathbf{y} = f_k(\mathbf{x}) = k(\mathbf{x} - x_0)^2 + s(\mathbf{x} - x_0) + y_{0k} \tag{1}$$

where $k$ controls curvature, $s$ is the skew, and $(x_0, y_0)$ is a translation vector. The parameters $k$, $s$, and $x_0$ are shared among all four curves, whereas each curve has its own vertical translation parameter $y_{0k}$. First the set of local maxima $U$ and minima $L$ of the vertical displacement are found. $x_0$ is determined by taking the average abscissa of extrema points. The lines of the model are then fitted to the extrema: the upper two lines to the maxima, and the lower two to the minima. The fit is performed using a probabilistic model for the extrema points given the lines. The idea is to find the line parameters $\theta^*$ that maximize the probability of

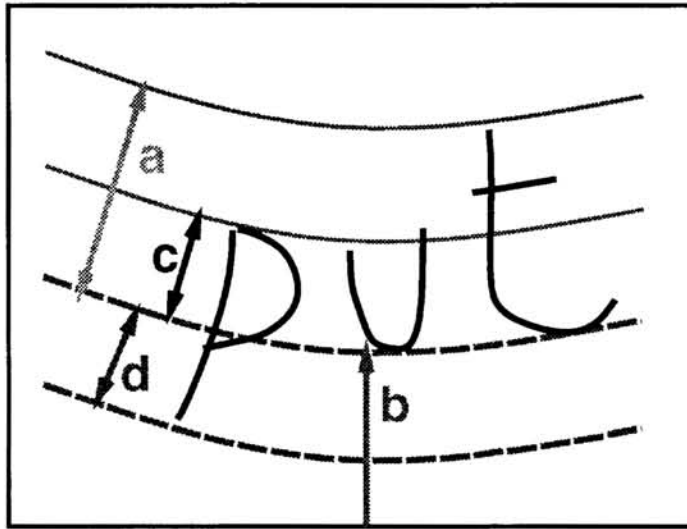

Figure 1: Word Normalization Model: Ascenders and core curves fit *y-maxima* whereas descenders and baseline curves fit *y-minima*. There are 6 parameters: **a** (ascenders curve height relative to baseline), **b** (baseline absolute vertical position), **c** (core line position), **d** (descenders curve position), **k** (curvature), **s** (angle).

generating the observed points.

$$\theta^* = \underset{\theta}{\mathrm{argmax}} \log P(X \mid \theta) + \log P(\theta) \tag{2}$$

The above conditional distribution is chosen to be a mixture of Gaussians (one per curve) whose means are the $y$-positions obtained from the actual $x$-positions through equation 1:

$$P(x_i, y_i \mid \theta) = \log \sum_{k=0}^{3} w_k N(y_i; f_k(x_i), \sigma_y) \tag{3}$$

where $N(x; \mu, \sigma)$ is a univariate Normal distribution of mean $\mu$ and standard deviation $\sigma$. The $w_k$ are the mixture parameters, some of which are set to 0 in order to constrain the upper (lower) points to be fitted to the upper (lower) curves. They are computed a-priori using measured frequencies of associations of extrema to curves on a large set of words. The priors $P(\theta)$ on the parameters are required to prevent the collapse of the curves. They can be used to incorporate a-priori information about the word geometry, such as the expected position of the baseline, or the height of the word. These priors for each parameter are chosen to be independent normal distributions whose standard deviations control the strength of the prior. The variables that associate each point with one of the curves are taken as hidden variables of the EM algorithm. One can thus derive an auxiliary function which can be analytically (and cheaply) solved for the 6 free parameters $\theta$. Convergence of the EM algorithm was typically obtained within 2 to 4 iterations (of maximization of the auxiliary function).

# 3   AMAP

The recognition of handwritten characters from a pen trajectory on a digitizing surface is often done in the time domain. Trajectories are normalized, and local

geometrical or dynamical features are sometimes extracted. The recognition is performed using curve matching (Tappert 90), or other classification techniques such as Neural Networks (Guyon et al 91). While, as stated earlier, these representations have several advantages, their dependence on stroke ordering and individual writing styles makes them difficult to use in high accuracy, writer independent systems that integrate the segmentation with the recognition.

Since the intent of the writer is to produce a legible *image*, it seems natural to preserve as much of the pictorial nature of the signal as possible, while at the same time exploit the sequential information in the trajectory. We propose a representation scheme, called AMAP, where pen trajectories are represented by low-resolution images in which each picture element contains information about the local properties of the trajectory. More generally, an AMAP can be viewed as a function in a multidimensional space where each dimension is associated with a local property of the trajectory, say the direction of motion $\theta$, the $X$ position, and the $Y$ position of the pen. The value of the function at a particular location $(\theta, X, Y)$ in the space represents a smooth version of the "density" of features in the trajectory that have values $(\theta, X, Y)$ (in the spirit of the generalized Hough transform). An AMAP is a multidimensional array (say 4x10x10) obtained by discretizing the feature density space into "boxes". Each array element is assigned a value equal to the integral of the feature density function over the corresponding box. In practice, an AMAP is computed as follows. At each sample on the trajectory, one computes the position of the pen $(X, Y)$ and orientation of the motion $\theta$ (and possibly other features, such as the local curvature $c$). Each element in the AMAP is then incremented by the amount of the integral over the corresponding box of a predetermined *point-spread function* centered on the coordinates of the feature vector. The use of a smooth point-spread function (say a Gaussian) ensures that smooth deformations of the trajectory will correspond to smooth transformations of the AMAP. An AMAP can be viewed as an "annotated image" in which each pixel is a feature vector.

A particularly useful feature of the AMAP representation is that it makes very few assumptions about the nature of the input trajectory. It does not depend on stroke ordering or writing speed, and it can be used with all types of handwriting (capital, lower case, cursive, punctuation, symbols). Unlike many other representations (such as global features), AMAPs can be computed for complete words without requiring segmentation.

## 4    Convolutional Neural Networks

Image-like representations such as AMAPs are particularly well suited for use in combination with Multi-Layer Convolutional Neural Networks (MLCNN) (Le Cun 89, Le Cun et al 90). MLCNNs are feed-forward neural networks whose architectures are tailored for minimizing the sensitivity to translations, rotations, or distortions of the input image. They are trained with a variation of the Back-Propagation algorithm (Rumelhart et al 86, Le Cun 86).

The units in MCLNNs are only connected to a local neighborhood in the previous layer. Each unit can be seen as a local feature detector whose function is determined by the learning procedure. Insensitivity to local transformations is built into the network architecture by constraining sets of units located at different places to use identical weight vectors, thereby forcing them to detect the same feature on different parts of the input. The outputs of the units at identical locations in different feature maps can be collectively thought of as a local feature vector. Features of increasing

complexity and globality are extracted by the neurons in the successive layers.

This weight-sharing technique has two interesting side effects. First, the number of free parameters in the system is greatly reduced since a large number of units share the same weights. Classically, MLCNNs are shown a single character at the input, and have a single set of outputs. However, an essential feature of MLCNNs is that they can be scanned (replicated) over large input fields containing multiple *unsegmented* characters (whole words) very economically by simply performing the convolutions on larger inputs. Instead of producing a single output vector, SDNNs produce a series of output vectors. The outputs detects and recognize characters at different (and overlapping) locations on the input. These multiple-input, multiple-output MLCNN are called Space Displacement Neural Networks (SDNN) (Matan et al 92).

One of the best networks we found for character recognition has 5 layers arranged as follows: layer 1: convolution with 8 kernels of size 3x3, layer 2: 2x2 subsampling, layer 3: convolution with 25 kernels of size 5x5, layer 4 convolution with 84 kernels of size 4x4, layer 5: 2x2 subsampling. The subsampling layers are essential to the network's robustness to distortions. The output layer is one (single MLCNN) or a series of (SDNN) 84-dimensional vectors. The target output configuration for each character class was chosen to be a *bitmap* of the corresponding character in a standard 7x12 (=84) pixel font. Such a code facilitates the correction of confusable characters by the postprocessor.

# 5   Post-Processing

The convolutional neural network can be used to give scores associated to characters when the network (or a piece of it corresponding to a single character output) has an input field, called a *segment*, that covers a connected subset of the whole word input. A *segmentation* is a sequence of such segments that covers the whole word input. Because there are in general many possible segmentations, sophisticated tools such as hidden Markov models and dynamic programming are used to search for the best segmentation.

In this paper, we consider two approaches to the segmentation problem called IN-SEG (for input segmentation) and OUTSEG (for output segmentation). The postprocessor can be generally decomposed into two levels: 1) character level scores and constraints obtained from the observations, 2) word level constraints (grammar, dictionary). The INSEG and OUTSEG systems share the second level.

In an INSEG system, the network is applied to a large number of heuristically segmented candidate characters. A *cutter* generates candidate *cuts*, which can potentially represent the boundary between two character segments. It also generates *definite cuts*, which we assume that no segment can cross. Using these, a number of candidate segments are constructed and the network is applied to each of them separately. Finally, for each high enough character score in each of the segment, a character hypothesis is generated, corresponding to a node in an *observation graph*. The connectivity and transition probabilities on the arcs of the observation graph represent segmentation and geometrical constraints (e.g., segments must not overlap and must cover the whole word, some transitions between characters are more or less likely given the geometrical relations between their images).

In an OUTSEG system, all segmentation decisions are delayed until after the recog-

nition, as is often done in speech recognition. The AMAP of the entire *word* is shown to an SDNN, which produces a sequence of output vectors equivalent to (but obtained much more cheaply than) scanning the single-character network over all possible pixel locations on the input. The Euclidean distances between each output vector and the targets are interpreted as log-likelihoods of the output given a class. To construct an *observation graph*, we use a set of character models (HMMs). Each character HMM models the sequence of network outputs observed for that character. We used three-state HMMs for each character, with a left and right state to model transitions and a center state for the character itself. The observation graph is obtained by connecting these character models, allowing any character to follow any character.

On top of the constraints given in the observation graph, additional constraints that are independent of the observations are given by what we call a *grammar graph*, which can embody lexical constraints. These constraints can be given in the form of a dictionary or of a character-level grammar (with transition probabilities), such as a trigram (in which we use the probability of observing a character in the context of the two previous ones). The recognition finds the best path in the observation graph that is compatible with the grammar graph. The INSEG and OUTSEG architecture are depicted in Figure 2.

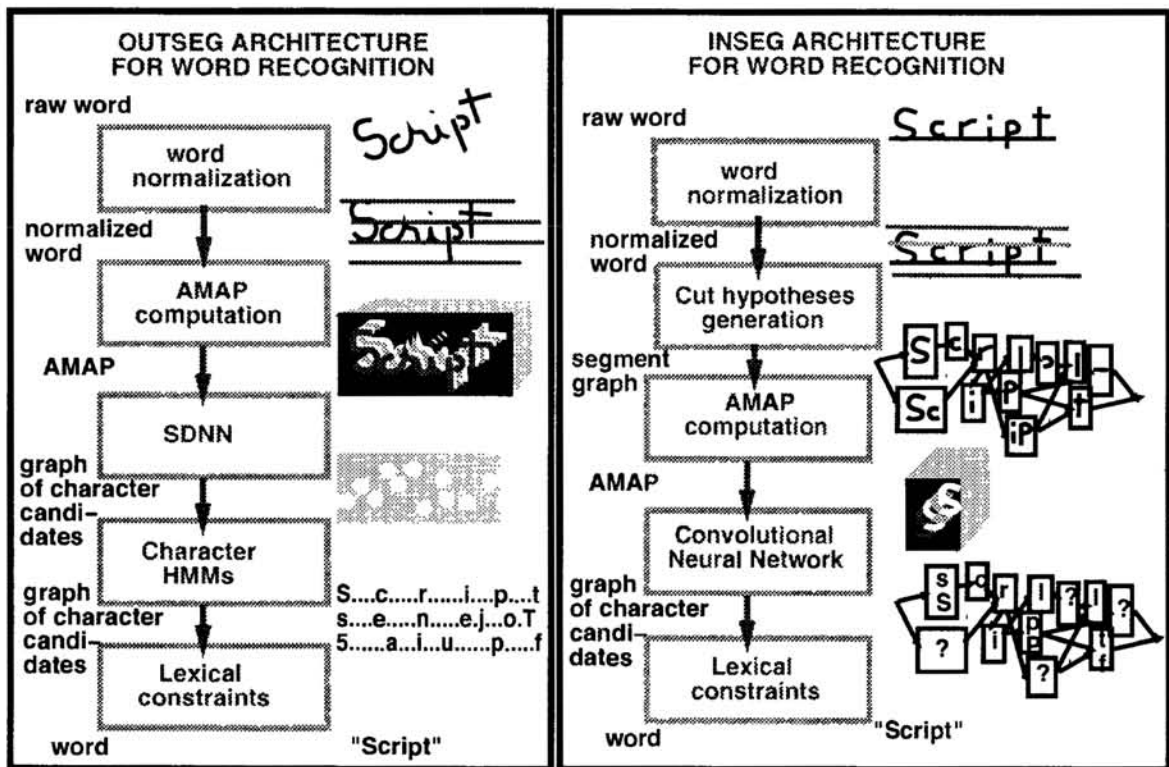

Figure 2: INSEG and OUTSEG architectures for word recognition.

A crucial contribution of our system is the joint training of the neural network and the post-processor with respect to a single criterion that approximates word-level errors. We used the following *discriminant* criterion: minimize the total cost (sum of negative log-likelihoods) along the "correct" paths (the ones that yield the correct interpretations), while minimizing the costs of *all* the paths (correct or not). The discriminant nature of this criterion can be shown with the following example. If

the cost of a path associated to the correct interpretation is much smaller than all other paths, then the criterion is very close to 0 and no gradient is back-propagated. On the other hand, if the lowest cost path yields an incorrect interpretation but differs from a path of correct interpretation on a sub-path, then very strong gradients will be propagated along that sub-path, whereas the other parts of the sequence will generate almost no gradient. Within a probabilistic framework, this criterion corresponds to the maximizing the mutual information (MMI) between the observations and the correct interpretation. During global training, it is optimized using (enhanced) stochastic gradient descent with respect to *all* the parameters in the system, most notably the network weights. Experiments described in the next section have shown important reductions in error rates when training with this word-level criterion instead of just training the network separately for each character. Similar combinations of neural networks with HMMs or dynamic programming have been proposed in the past, for speech recognition problems (Bengio et al 92).

## 6    Experimental Results

In a first set of experiments, we evaluated the generalization ability of the neural network classifier coupled with the word normalization preprocessing and AMAP input representation. All results are in *writer independent* mode (different writers in training and testing). Tests on a database of isolated characters were performed separately on four types of characters: upper case (2.99% error on 9122 patterns), lower case (4.15% error on 8201 patterns), digits (1.4% error on 2938 patterns), and punctuation (4.3% error on 881 patterns). Experiments were performed with the network architecture described above.

The second and third set of experiments concerned the recognition of lower case words (writer independent). The tests were performed on a database of 881 words. First we evaluated the improvements brought by the word normalization to the INSEG system. For the OUTSEG system we have to use a word normalization since the network sees a whole word at a time. With the INSEG system, and before doing any word-level training, we obtained without word normalization 7.3% and 3.5% word and character errors (adding insertions, deletions and substitutions) when the search was constrained within a 25461-word dictionary. When using the word normalization preprocessing instead of a character level normalization, error rates dropped to 4.6% and 2.0% for word and character errors respectively, i.e., a relative drop of 37% and 43% in word and character error respectively.

In the third set of experiments, we measured the improvements obtained with the joint training of the neural network and the post-processor with the word-level criterion, in comparison to training based only on the errors performed at the character level. Training was performed with a database of 3500 lower case words. For the OUTSEG system, without any dictionary constraints, the error rates dropped from 38% and 12.4% word and character error to 26% and 8.2% respectively after word-level training, i.e., a relative drop of 32% and 34%. For the INSEG system and a slightly improved architecture, without any dictionary constraints, the error rates dropped from 22.5% and 8.5% word and character error to 17% and 6.3% respectively, i.e., a relative drop of 24.4% and 25.6%. With a 25461-word dictionary, errors dropped from 4.6% and 2.0% word and character errors to 3.2% and 1.4% respectively after word-level training, i.e., a relative drop of 30.4% and 30.0%. Finally, some further improvements can be obtained by drastically reducing the size of the dictionary to 350 words, yielding 1.6% and 0.94% word and character errors.

## 7   Conclusion

We have demonstrated a new approach to on-line handwritten word recognition that uses word or sentence-level preprocessing and normalization, image-like representations, Convolutional neural networks, word models, and global training using a highly discriminant word-level criterion. Excellent accuracy on various writer independent tasks were obtained with this combination.

**References**

Bengio, Y., R. De Mori and G. Flammia and R. Kompe. 1992. Global Optimization of a Neural Network-Hidden Markov Model Hybrid. *IEEE Transactions on Neural Networks* v.3, nb.2, pp.252–259.

Burges, C., O. Matan, Y. Le Cun, J. Denker, L. Jackel, C. Stenard, C. Nohl and J. Ben. 1992. Shortest Path Segmentation: A Method for Training a Neural Network to Recognize character Strings. Proc. IJCNN'92 (Baltimore), pp. 165–172, v.3.

Guyon, I., Albrecht, P., Le Cun, Y., Denker, J. S., and Weissman, H. 1991 design of a neural network character recognizer for a touch terminal. *Pattern Recognition*, 24(2):105–119.

Le Cun, Y. 1986. Learning Processes in an Asymmetric Threshold Network. In Bienenstock, E., Fogelman-Soulié, F., and Weisbuch, G., editors, *Disordered systems and biological organization*, pages 233–240, Les Houches, France. Springer-Verlag.

Le Cun, Y. 1989. Generalization and Network Design Strategies. In Pfeifer, R., Schreter, Z., Fogelman, F., and Steels, L., editors, *Connectionism in Perspective*, Zurich, Switzerland. Elsevier. an extended version was published as a technical report of the University of Toronto.

Le Cun, Y., Matan, O., Boser, B., Denker, J. S., Henderson, D., Howard, R. E., Hubbard, W., Jackel, L. D., and Baird, H. S. 1990. Handwritten Zip Code Recognition with Multilayer Networks. In IAPR, editor, *Proc. of the International Conference on Pattern Recognition*, Atlantic City. IEEE.

Matan, O., Burges, C. J. C., LeCun, Y., and Denker, J. S. 1992. Multi-Digit Recognition Using a Space Displacement Neural Network. In Moody, J. M., Hanson, S. J., and Lippman, R. P., editors, *Neural Information Processing Systems*, volume 4. Morgan Kaufmann Publishers, San Mateo, CA.

Rumelhart, D. E., Hinton, G. E., and Williams, R. J. 1986. Learning internal representations by error propagation. In *Parallel distributed processing: Explorations in the microstructure of cognition*, volume I, pages 318–362. Bradford Books, Cambridge, MA.

Schenkel, M., Guyon, I., Weissman, H., and Nohl, C. 1993. TDNN Solutions for Recognizing On-Line Natural Handwriting. In *Advances in Neural Information Processing Systems 5*. Morgan Kaufman.

Tappert, C., Suen, C., and Wakahara, T. 1990. The state of the art in on-line handwriting recognition. *IEEE Trans. PAMI*, 12(8).

## Footnotes

*also, AT&T Bell Labs, Holmdel NJ 07733
